# Memory-Based Methods for Regression and Classification

**Thomas G. Dietterich** and **Dietrich Wettschereck**
Department of Computer Science
Oregon State University
Corvallis, OR 97331-3202

**Chris G. Atkeson**
MIT AI Lab
545 Technology Square
Cambridge, MA 02139

**Andrew W. Moore**
School of Computer Science
Carnegie Mellon University
Pittsburgh, PA 15213

Memory-based learning methods operate by storing all (or most) of the training data and deferring analysis of that data until "run time" (i.e., when a query is presented and a decision or prediction must be made). When a query is received, these methods generally answer the query by retrieving and analyzing a small subset of the training data—namely, data in the immediate neighborhood of the query point. In short, memory-based methods are "lazy" (they wait until the query) and "local" (they use only a local neighborhood). The purpose of this workshop was to review the state-of-the-art in memory-based methods and to understand their relationship to "eager" and "global" learning algorithms such as batch backpropagation.

There are two essential components to any memory-based algorithm: the method for defining the "local neighborhood" and the learning method that is applied to the training examples in the local neighborhood.

We heard several talks on issues related to defining the "local neighborhood". Federico Girosi and Trevor Hastie reviewed "kernel" methods in classification and regression. A kernel function $K(d)$ maps the distance $d$ from the query point to a training example into a real value. In the well-known Parzen window approach, the kernel is a fixed-width gaussian, and a new example is classified by taking a weighted vote of the classes of all training examples, where the weights are determined by the gaussian kernel. Because of the "local" shape of the gaussian, distant training examples have essentially no influence on the classification decision. In regression problems, a common approach is to construct a linear regression fit to the data, where the squared error from each data point is weighted by the kernel.

Hastie described the kernel used in the LOESS method: $K(d) = (1-d^3)^3$ ($0 \le d \le 1$ and $K(d) = 0$ otherwise). To adapt to the local density of training examples, this kernel is scaled to cover the $k$th nearest neighbor. Many other kernels have been explored, with particular attention to bias and variance at the extremes of the

training data. Methods have been developed for computing the effective number of parameters used by these kernel methods.

Girosi pointed out that some "global" learning algorithms (e.g., splines) are equivalent to kernel methods. The kernels often have informative shapes. If a kernel places most weight near the query point, then we can say that the learning algorithm is local, even if the algorithm performs a global analysis of the training data at learning time. An open problem is to determine whether multi-layer sigmoidal networks have equivalent kernels and, if so, what their shapes are.

David Lowe described a classification algorithm based on gaussian kernels. The kernel is scaled by the mean distance to the $k$ nearest neighbors. His Variable-kernel Similarity Metric (VSM) algorithm learns the weights of a weighted Euclidean distance in order to maximize the leave-one-out accuracy of the algorithm. Excellent results have been obtained on benchmark tasks (e.g., NETtalk).

Patrice Simard described the tangent distance method. In optical character recognition, the features describing a character change as that character is rotated, translated, or scaled. Hence, each character actually corresponds to a manifold of points in feature space. The tangent distance is a planar approximation to the distance between two manifolds (for two characters). Using tangent distance with the nearest neighbor rule gives excellent results in a zipcode recognition task.

Leon Bottou also employed a sophisticated distance metric by using the Euclidean distance between the hidden unit activations of the final hidden layer in the Bell Labs "LeNet" character recognizer. A simple linear classifier (with weight decay) was constructed to classify each query. Bottou also showed that there is a tradeoff between the quality of the distance metric and the locality of the learning algorithm. The tangent distance is a near-perfect metric, and it can use the highly local first-nearest-neighbor rule. The hidden layer of the LeNet gives a somewhat better metric, but it requires approximately 200 "local" examples. With the raw features, LeNet itself requires all of the training examples.

We heard several talks on methods that are local but not lazy. John Platt described his RAN (Resource Allocating Network) that learns a linear combination of radial basis functions by iterative training on the data. Bernd Fritzke described his improvements to RAN. Stephen Omohundro explained model merging, which initially learns local patches and, when the data justifies, combines primitive patches into larger high-order patches. Dietrich Wettschereck presented BNGE, which learns a set of local axis-parallel rectangular patches.

Finally, Andrew Moore, Chris Atkeson, and Stefan Schaal described integrated memory-based learning systems for control applications. Moore's system applies huge amounts of cross-validation to select distance metrics, kernels, kernel widths, and so on. Atkeson advocated *radical localism*—all algorithm parameters should be determined by lazy, local methods. He described algorithms for obtaining confidence intervals on the outputs of local regression as well as techniques for outlier removal. One method seeks to minimize the width of the confidence intervals.

Some of the questions left unanswered by the workshop include these: Are there inherent computational penalties that lazy methods must pay (but eager methods can avoid)? How about the reverse? For what problems are local methods appropriate?